# Laplacian Eigenmaps and Spectral Techniques for Embedding and Clustering

**Mikhail Belkin and Partha Niyogi**
Depts. of Mathematics and Computer Science
The University of Chicago
Hyde Park, Chicago, IL 60637.
(misha@math.uchicago.edu,niyogi@cs.uchicago.edu)

## Abstract

Drawing on the correspondence between the graph Laplacian, the Laplace-Beltrami operator on a manifold, and the connections to the heat equation, we propose a geometrically motivated algorithm for constructing a representation for data sampled from a low dimensional manifold embedded in a higher dimensional space. The algorithm provides a computationally efficient approach to non-linear dimensionality reduction that has locality preserving properties and a natural connection to clustering. Several applications are considered.

In many areas of artificial intelligence, information retrieval and data mining, one is often confronted with intrinsically low dimensional data lying in a very high dimensional space. For example, gray scale $n \times n$ images of a fixed object taken with a moving camera yield data points in $\mathbb{R}^{n^2}$. However, the intrinsic dimensionality of the space of all images of the same object is the number of degrees of freedom of the camera – in fact the space has the natural structure of a manifold embedded in $\mathbb{R}^{n^2}$. While there is a large body of work on dimensionality reduction in general, most existing approaches do not explicitly take into account the structure of the manifold on which the data may possibly reside. Recently, there has been some interest (Tenenbaum et al, 2000; Roweis and Saul, 2000) in the problem of developing low dimensional representations of data in this particular context. In this paper, we present a new algorithm and an accompanying framework of analysis for geometrically motivated dimensionality reduction.

The core algorithm is very simple, has a few local computations and one sparse eigenvalue problem. The solution reflects the intrinsic geometric structure of the manifold. The justification comes from the role of the Laplacian operator in providing an optimal embedding. The Laplacian of the graph obtained from the data points may be viewed as an approximation to the Laplace-Beltrami operator defined on the manifold. The embedding maps for the data come from approximations to a natural map that is defined on the entire manifold. The framework of analysis

presented here makes this connection explicit. While this connection is known to geometers and specialists in spectral graph theory (for example, see [1, 2]) to the best of our knowledge we do not know of any application to data representation yet. The connection of the Laplacian to the heat kernel enables us to choose the weights of the graph in a principled manner.

The locality preserving character of the Laplacian Eigenmap algorithm makes it relatively insensitive to outliers and noise. A byproduct of this is that the algorithm implicitly emphasizes the natural clusters in the data. Connections to spectral clustering algorithms developed in learning and computer vision (see Shi and Malik, 1997) become very clear. Following the discussion of Roweis and Saul (2000), and Tenenbaum et al (2000), we note that the biological perceptual apparatus is confronted with high dimensional stimuli from which it must recover low dimensional structure. One might argue that if the approach to recovering such low-dimensional structure is inherently local, then a natural clustering will emerge and thus might serve as the basis for the development of categories in biological perception.

# 1  The Algorithm

Given $k$ points $\mathbf{x}_1, \ldots, \mathbf{x}_k$ in $\mathbb{R}^l$, we construct a weighted graph with $k$ nodes, one for each point, and the set of edges connecting neighboring points to each other.

1. Step 1. [*Constructing the Graph*] We put an edge between nodes $i$ and $j$ if $x_i$ and $x_j$ are "close". There are two variations:

   (a) $\epsilon$-neighborhoods. [*parameter* $\epsilon \in \mathbb{R}$] Nodes $i$ and $j$ are connected by an edge if $\|\mathbf{x}_i - \mathbf{x}_j\|^2 < \epsilon$.
   Advantages: geometrically motivated, the relationship is naturally symmetric.
   Disadvantages: often leads to graphs with several connected components, difficult to choose $\epsilon$.

   (b) $n$ nearest neighbors. [*parameter* $n \in \mathbb{N}$] Nodes $i$ and $j$ are connected by an edge if $i$ is among $n$ nearest neighbors of $j$ or $j$ is among $n$ nearest neighbors of $i$.
   Advantages: simpler to choose, tends to lead to connected graphs.
   Disadvantages: less geometrically intuitive.

2. Step 2. [Choosing the weights] Here as well we have two variations for weighting the edges:

   (a) Heat kernel. [*parameter* $t \in \mathbb{R}$]. If nodes $i$ and $j$ are connected, put

   $$W_{ij} = e^{-\frac{\|\mathbf{x}_i - \mathbf{x}_j\|^2}{t}}$$

   The justification for this choice of weights will be provided later.

   (b) Simple-minded. [*No parameters*]. $W_{ij} = 1$ if and only if vertices $i$ and $j$ are connected by an edge.
   A simplification which avoids the necessity of choosing $t$.

3. Step 3. [Eigenmaps] Assume the graph $G$, constructed above, is connected, otherwise proceed with Step 3 for each connected component.

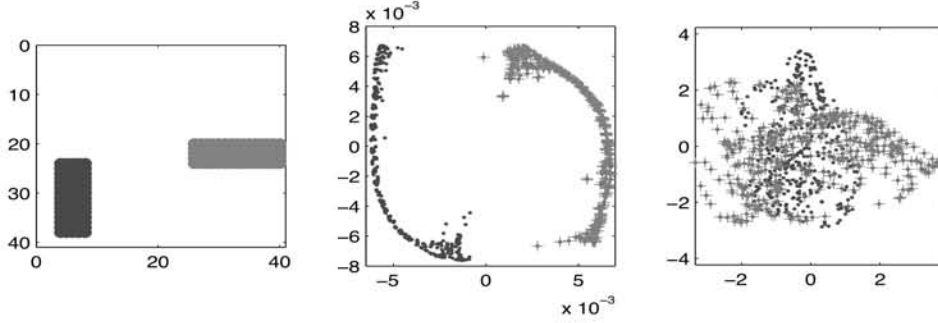

Figure 1: The left panel shows a horizontal and a vertical bar. The middle panel is a two dimensional representation of the set of all images using the Laplacian eigenmaps. The right panel shows the result of a principal components analysis using the first two principal directions to represent the data. Dots correspond to vertical bars and '+' signs correspond to horizontal bars.

Compute eigenvalues and eigenvectors for the generalized eigenvector problem:

$$L\mathbf{y} = \lambda D\mathbf{y} \tag{1}$$

where $D$ is diagonal weight matrix, its entries are column (or row, since $W$ is symmetric) sums of $W$, $D_{ii} = \sum_j W_{ji}$. $L = D - W$ is the Laplacian matrix. Laplacian is a symmetric, positive semidefinite matrix which can be thought of as an operator on functions defined on vertices of $G$.

Let $\mathbf{y}_0, \ldots, \mathbf{y}_{k-1}$ be the solutions of equation 1, ordered according to their eigenvalues with $\mathbf{y}_0$ having the smallest eigenvalue (in fact 0). The image of $\mathbf{x}_i$ under the embedding into the lower dimensional space $\mathbb{R}^m$ is given by $(\mathbf{y}_1(i), \ldots, \mathbf{y}_m(i))$.

## 2 Justification

Recall that given a data set we construct a weighted graph $G = (V, E)$ with edges connecting nearby points to each other. Consider the problem of mapping the weighted connected graph $G$ to a line so that connected points stay as close together as possible. We wish to choose $y_i \in \mathbb{R}$ to minimize

$$\sum_{i,j} (y_i - y_j)^2 W_{ij}$$

under appropriate constraints. Let $\mathbf{y} = (y_1, y_2, \ldots, y_n)^T$ be the map from the graph to the real line. First, note that for any $\mathbf{y}$, we have

$$\frac{1}{2} \sum_{i,j} (y_i - y_j)^2 W_{ij} = \mathbf{y}^T L \mathbf{y} \tag{2}$$

where as before, $L = D - W$. To see this, notice that $W_{ij}$ is symmetric and $D_{ii} = \sum_j W_{ij}$. Thus $\sum_{i,j} (y_i - y_j)^2 W_{ij}$ can be written as

$$\sum_{i,j} (y_i^2 + y_j^2 - 2y_i y_j) W_{ij} = \sum_i y_i^2 D_{ii} + \sum_j y_j^2 D_{jj} - 2 \sum_{i,j} y_i y_j W_{ij} = 2\mathbf{y}^T L \mathbf{y}$$

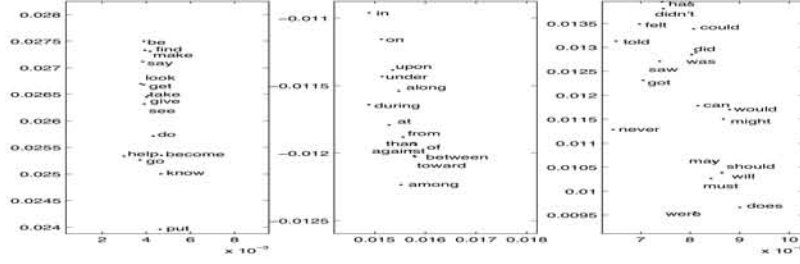

Figure 3: Fragments labeled by arrows in figure 2, from left to right. The first contains infinitives of verbs, the second contains prepositions and the third mostly modal and auxiliary verbs. We see that syntactic structure is well-preserved.

Therefore, the minimization problem reduces to finding $\text{argmin}_{\mathbf{y}^T D \mathbf{y}=1} \mathbf{y}^T L \mathbf{y}$.

The constraint $\mathbf{y}^T D \mathbf{y} = 1$ removes an arbitrary scaling factor in the embedding. Matrix $D$ provides a natural measure on the vertices of the graph. From eq. 2, we see that $L$ is a positive semidefinite matrix and the vector $\mathbf{y}$ that minimizes the objective function is given by the minimum eigenvalue solution to the generalized eigenvalue problem $L\mathbf{y} = \lambda D \mathbf{y}$.

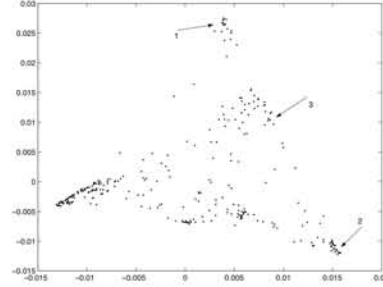

Figure 2: 300 most frequent words of the Brown corpus represented in the spectral domain.

Let $\mathbf{1}$ be the constant function taking value 1 at each vertex. It is easy to see that $\mathbf{1}$ is an eigenvector with eigenvalue 0. If the graph is connected, $\mathbf{1}$ is the only eigenvector for $\lambda = 0$. To eliminate this trivial solution which collapses all vertices of $G$ onto the real number 1, we put an additional constraint of orthogonality to obtain

$$\mathbf{y}_{opt} = \text{argmin}_{\substack{\mathbf{y}^T D \mathbf{y}=1 \\ \mathbf{y}^T D \mathbf{1}=0}} \mathbf{y}^T L \mathbf{y}$$

Thus, the solution $\mathbf{y}_{opt}$ is now given by the eigenvector with the smallest non-zero eigenvalue. More generally, the embedding of the graph into $\mathbb{R}^m$ ($m > 1$) is given by the $n \times m$ matrix $Y = [\mathbf{y}_1 \mathbf{y}_2 \ldots \mathbf{y}_m]$ where the $i$th row, denoted by $Y_i^T$, provides the embedding coordinates of the $i$th vertex. Thus we need to minimize

$$\sum_{i,j} ||Y_i - Y_j||^2 W_{ij} = \text{tr}(Y^T L Y)$$

This reduces now to

$$Y_{opt} = \text{argmin}_{Y^T D Y=I} \text{tr}(Y^T L Y)$$

For the one-dimensional embedding problem, the constraint prevents collapse onto a point. For the $m$-dimensional embedding problem, the constraint presented above prevents collapse onto a subspace of dimension less than $m$.

## 2.1 The Laplace-Beltrami Operator

The Laplacian of a graph is analogous to the Laplace-Beltrami operator on manifolds.

Consider a smooth $m$-dimensional manifold $\mathcal{M}$ embedded in $\mathbb{R}^k$. The Riemannian structure (metric tensor) on the manifold is induced by the standard Riemannian structure on $\mathbb{R}^k$. Suppose we have a map $f : \mathcal{M} \to \mathbb{R}$. The gradient $\nabla f(x)$ (which in local coordinates can be written as $\nabla f(x) = \sum_{i=1}^{n} \frac{\partial f}{\partial x_i} \partial_{x_i}$) is a vector field on the manifold, such that for small $\delta x$ (in a local coordinate chart)

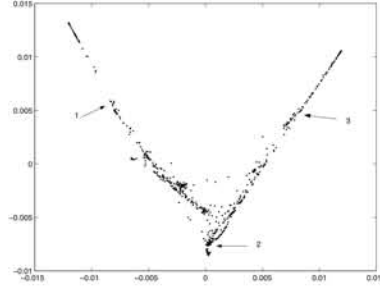

Figure 4: 685 speech datapoints plotted in the two dimensional Laplacian spectral representation.

$$|f(x + \delta x) - f(x)| \approx |\langle \nabla f(x), \delta x \rangle| \leq ||\nabla f|| \, ||\delta x||$$

Thus we see that if $||\nabla f||$ is small, points near $x$ will be mapped to points near $f(x)$. We therefore look for a map that best preserves locality on average by trying to find

$$\text{argmin}_{||f||_{L^2(\mathcal{M})}=1} \int_{\mathcal{M}} ||\nabla f(x)||^2$$

Minimizing $\int_{\mathcal{M}} ||\nabla f(x)||^2$ corresponds directly to minimizing $Lf = \frac{1}{2} \sum_{i,j} (f_i - f_j)^2 W_{ij}$ on a graph. Minimizing the squared gradient reduces to finding eigenfunctions of the Laplace-Beltrami operator $\mathcal{L}$. Recall that $\mathcal{L} \overset{def}{=} \text{div} \nabla (f)$, where div is the divergence. It follows from the Stokes theorem that $-\text{div}$ and $\nabla$ are formally adjoint operators, i.e. if $f$ is a function and $\mathbf{X}$ is a vector field $\int_{\mathcal{M}} \langle \mathbf{X}, \nabla f \rangle = \int_{\mathcal{M}} \text{div}(\mathbf{X}) f$. Thus

$$\int_{\mathcal{M}} ||\nabla f||^2 = \int_{\mathcal{M}} \mathcal{L}(f) f$$

We see that $\mathcal{L}$ is positive semidefinite and the $f$ that minimizes $\int_{\mathcal{M}} ||\nabla f||^2$ has to be an eigenfunction of $\mathcal{L}$.

## 2.2 Heat Kernels and the Choice of Weight Matrix

The Laplace-Beltrami operator on differentiable functions on a manifold $\mathcal{M}$ is intimately related to the heat flow. Let $f : \mathcal{M} \to \mathbb{R}$ be the initial heat distribution, $u(x,t)$ be the heat distribution at time $t$ ($u(x,0) = f(x)$). The heat

equation is the partial differential equation $\frac{\partial u}{\partial t} = \mathcal{L}u$. The solution is given by $u(x,t) = \int_{\mathcal{M}} H_t(x,y)f(y)$ where $H_t$ is the heat kernel – the Green's function for this PDE. Therefore,

$$\mathcal{L}f(x) = \mathcal{L}u(x,0) = \left( \frac{\partial}{\partial t} \left[ \int_{\mathcal{M}} H_t(x,y)f(y) \right] \right)_{t=0}$$

Locally, the heat kernel is approximately equal to the Gaussian, $H_t(x,y) \approx (4\pi t)^{-\frac{n}{2}} e^{-\frac{\|x-y\|^2}{4t}}$ where $\|x-y\|$ ($x$ and $y$ are in local coordinates) and $t$ are both sufficiently small and $n = dim\,\mathcal{M}$. Notice that as $t$ tends to 0, the heat kernel $H_t(x,y)$ becomes increasingly localized and tends to Dirac's $\delta$-function, i.e., $\lim_{t \to 0} \int_{\mathcal{M}} H_t(x,y)f(y) = f(x)$. Therefore, for small $t$ from the definition of the derivative we have

$$\mathcal{L}f(\mathbf{x}_i) \approx -\frac{1}{t} \left[ f(x) - (4\pi t)^{-\frac{n}{2}} \int_{\mathcal{M}} e^{-\frac{\|x-y\|^2}{4t}} f(y)dy \right]$$

If $\mathbf{x}_1, \ldots, \mathbf{x}_k$ are data points on $\mathcal{M}$, the last expression can be approximated by

$$\mathcal{L}f(\mathbf{x}_i) = -\frac{1}{t} \left[ f(\mathbf{x}_i) - \frac{1}{k}(4\pi t)^{-\frac{n}{2}} \sum_{\substack{\mathbf{x}_j \\ 0 < \|\mathbf{x}_j - \mathbf{x}_i\| < \epsilon}} e^{-\frac{\|\mathbf{x}_i - \mathbf{x}_j\|^2}{4t}} f(\mathbf{x}_j) \right]$$

The coefficient $\frac{1}{t}$ is global and will not affect the eigenvectors of the discrete Laplacian. Since the inherent dimensionality of $\mathcal{M}$ may be unknown, we put $\alpha = \frac{1}{k}(4\pi t)^{\frac{n}{2}}$. Noticing that the Laplacian of the constant function is zero, we immediately have $\frac{1}{\alpha} = \sum_{\substack{\mathbf{x}_j \\ 0 < \|\mathbf{x}_j - \mathbf{x}_i\| < \epsilon}} e^{-\frac{\|\mathbf{x}_i - \mathbf{x}_j\|^2}{4t}}$. Notice, however, that we do not have to worry about $\alpha$, since the graph Laplacian $L$ will choose the correct multiplier for us. Finally we see how to choose the edge weights for the adjacency matrix $W$:

$$W_{ij} = \begin{cases} e^{-\frac{\|\mathbf{x}_i - \mathbf{x}_j\|^2}{4t}} & \text{if } \|\mathbf{x}_i - \mathbf{x}_j\| < \epsilon \\ 0 & \text{otherwise} \end{cases}$$

## 3  Examples

**Example 1 – A Toy Vision Example:** Consider binary images of vertical and horizontal bars located at arbitrary points in the $40 \times 40$ visual field. We choose 1000 images, each containing either a vertical or a horizontal bar (500 containing vertical bars and 500 horizontal bars) at random. Fig. 1 shows the result of applying the Laplacian Eigenmaps compared to PCA.

**Example 2 – Words in the Brown Corpus:** Fig. 2 shows the results of an experiment conducted with the 300 most frequent words in the Brown corpus – a collection of texts containing about a million words available in electronic format. Each word is represented as a vector in a 600 dimensional space using information about the frequency of its left and right neighbors (computed from the bigram statistics of the corpus).

**Example 3 – Speech:** In Fig. 4 we consider the low dimensional representations arising from applying the Laplacian Eigenmap algorithm to a sentence of speech

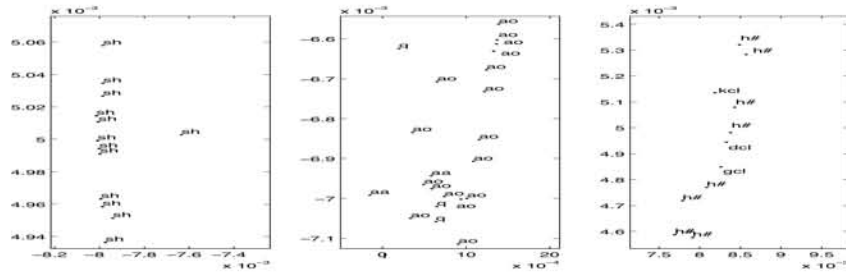

Figure 5: A blowup of the three selected regions in figure 4, from left to right. Notice the phonetic homogeneity of the chosen regions. Note that points marked with the same symbol may arise from occurrences of the same phoneme at different points in the utterance. The symbol "sh" stands for the fricative in the word *she*; "aa","ao" stand for vowels in the words *dark* and *all* respectively; "kcl","dcl","gcl" stand for closures preceding the stop consonants "k","d","g" respectively. "h#" stands for silence.

sampled at 1kHz. Short-time Fourier spectra were computed at 5 ms intervals yielding 685 vectors of 256 Fourier coefficients for every 30 ms chunk of the speech signal. Each vector is labeled according to the identity of the phonetic segment it belonged to. Fig. 4 shows the speech data points plotted in the two dimensional Laplacian representation. The two "spokes" correspond predominantly to fricatives and closures respectively. The central portion corresponds mostly to periodic sounds like vowels, nasals, and semivowels. Fig. 5 shows three different regions of the representation space.

## References

[1] Fan R. K. Chung, *Spectral Graph Theory*, Regional Conference Series in Mathematics, number 92, 1997

[2] Fan R. K. Chung, A. Grigor'yan, S.-T. Yau, *Higher eigenvalues and isoperimetric inequalities on Riemannian manifolds and graphs*, Communications on Analysis and Geometry, to appear,

[3] S. Rosenberg, *The Laplacian on a Riemmannian Manifold*, Cambridge University Press, 1997,

[4] Sam T. Roweis, Lawrence K. Saul, *Nonlinear Dimensionality Reduction by Locally Linear Embedding*, Science, vol 290, 22 Dec. 2000,

[5] Jianbo Shi, Jitendra Malik, *Normalized Cuts and Image Segmentation*, IEEE Transactions on PAMI, vol 22, no 8, August 2000

[6] J. B. Tenenbaum, V. de Silva, J. C. Langford, *A Global Geometric Framework for Nonlinear Dimensionality Reduction*, Science, Vol 290, 22 Dec. 2000
